# Unmixing Hyperspectral Data

**Lucas Parra, Clay Spence, Paul Sajda**
Sarnoff Corporation, CN-5300, Princeton, NJ 08543, USA
{*lparra,cspence,psajda*} *@sarnoff.com*

**Andreas Ziehe, Klaus-Robert Müller**
GMD FIRST.IDA, Kekuléstr. 7, 12489 Berlin, Germany
{*ziehe,klaus*} *@first.gmd.de*

## Abstract

In hyperspectral imagery one pixel typically consists of a mixture of the reflectance spectra of several materials, where the mixture coefficients correspond to the abundances of the constituting materials. We assume linear combinations of reflectance spectra with some additive normal sensor noise and derive a probabilistic MAP framework for analyzing hyperspectral data. As the material reflectance characteristics are not know a priori, we face the problem of unsupervised linear unmixing. The incorporation of different prior information (e.g. positivity and normalization of the abundances) naturally leads to a family of interesting algorithms, for example in the noise-free case yielding an algorithm that can be understood as constrained independent component analysis (ICA). Simulations underline the usefulness of our theory.

## 1 Introduction

Current hyperspectral remote sensing technology can form images of ground surface reflectance at a few hundred wavelengths simultaneously, with wavelengths ranging from 0.4 to 2.5 $\mu$m and spatial resolutions of 10-30 m. The applications of this technology include environmental monitoring and mineral exploration and mining. The benefit of hyperspectral imagery is that many different objects and terrain types can be characterized by their spectral signature.

The first step in most hyperspectral image analysis systems is to perform a spectral unmixing to determine the original spectral signals of some set of prime materials. The basic difficulty is that for a given image pixel the spectral reflectance patterns of the surface materials is in general not known a priori. However there are general physical and statistical priors which can be exploited to potentially improve spectral unmixing. In this paper we address the problem of unmixing hyperspectral imagery through incorporation of physical and statistical priors within an unsupervised Bayesian framework.

We begin by first presenting the linear superposition model for the reflectances measured. We then discuss the advantages of unsupervised over supervised systems.

We derive a general maximum a posteriori (MAP) framework to find the material spectra and infer the abundances. Interestingly, depending on how the priors are incorporated, the zero noise case yields (i) a simplex approach or (ii) a constrained ICA algorithm. Assuming non-zero noise our MAP estimate utilizes a constrained least squares algorithm. The two latter approaches are new algorithms whereas the simplex algorithm has been previously suggested for the analysis of hyperspectral data.

**Linear Modeling** To a first approximation the intensities $\mathbf{X}$ $(x_{i\lambda})$ measured in each spectral band $\lambda = 1, \ldots, L$ for a given pixel $i = 1, \ldots, N$ are linear combinations of the reflectance characteristics $\mathbf{S}$ $(s_{m\lambda})$ of the materials $m = 1, \ldots, M$ present in that area. Possible errors of this approximation and sensor noise are taken into account by adding a noise term $\mathbf{N}$ $(n_{i\lambda})$. In matrix form this can be summarized as

$$\mathbf{X} = \mathbf{AS} + \mathbf{N}, \text{ subject to: } \mathbf{A1}_M = \mathbf{1}_L, \quad \mathbf{A} \geq \mathbf{0}, \tag{1}$$

where matrix $\mathbf{A}$ $(a_{im})$ represents the abundance of material $m$ in the area corresponding to pixel $i$, with positivity and normalization constraints. Note that ground inclination or a changing viewing angle may cause an overall scale factor for all bands that varies with the pixels. This can be incorporated in the model by simply replacing the constraint $\mathbf{A1}_M = \mathbf{1}_L$ with $\mathbf{A1}_M \leq \mathbf{1}_L$ which does does not affect the discussion in the remainder of the paper. This is clearly a simplified model of the physical phenomena. For example, with spatially fine grained mixtures, called *intimate mixtures*, multiple reflectance may causes departures from this first order model. Additionally there are a number of inherent spatial variations in real data, such as inhomogeneous vapor and dust particles in the atmosphere, that will cause a departure from the linear model in equation (1). Nevertheless, in practical applications a linear model has produced reasonable results for *areal mixtures*.

**Supervised vs. Unsupervised techniques** *Supervised* spectral unmixing relies on the prior knowledge about the reflectance patterns $\mathbf{S}$ of candidate surface materials, sometimes called *endmembers*, or expert knowledge and a series of semi-automatic steps to find the constituting materials in a particular scene. Once the user identifies a pixel $i$ containing a single material, i.e. $a_{im} = 1$ for a given $m$ and $i$, the corresponding spectral characteristics of that material can be taken directly from the observations, i.e., $s_{m\lambda} = x_{i\lambda}$ [4]. Given knowledge about the endmembers one can simply find the abundances by solving a constrained least squares problem. The problem with such supervised techniques is that finding the correct $\mathbf{S}$ may require substantial user interaction and the result may be error prone, as a pixel that actually contains a mixture can be misinterpreted as a pure endmember. Another approach obtains endmembers directly from a database. This is also problematic because the actual surface material on the ground may not match the database entries, due to atmospheric absorption or other noise sources. Finding close matches is an ambiguous process as some endmembers have very similar reflectance characteristics and may match several entries in the database.

*Unsupervised* unmixing, in contrast, tries to identify the endmembers and mixtures directly from the observed data $\mathbf{X}$ without any user interaction. There are a variety of such approaches. In one approach a simplex is fit to the data distribution [7, 6, 2]. The resulting vertex points of the simplex represent the desired endmembers, but this technique is very sensitive to noise as a few boundary points can potentially change the location of the simplex vertex points considerably. Another approach by Szu [9] tries to find abundances that have the highest entropy subject to constraints that the amount of materials is as evenly distributed as possible – an assumption

which is clearly not valid in many actual surface material distributions. A relatively new approach considers modeling the statistical information across wavelength as statistically independent AR processes [1]. This leads directly to the contextual linear ICA algorithm [5]. However, the approach in [1] does not take into account constraints on the abundances, noise, or prior information. Most importantly, the method [1] can only integrate information from a small number of pixels at a time (same as the number of endmembers). Typically however we will have only a few endmembers but many thousand pixels.

## 2    The Maximum A Posterior Framework

### 2.1    A probabilistic model of unsupervised spectral unmixing

Our model has observations or data $\mathbf{X}$ and hidden variables $\mathbf{A}$, $\mathbf{S}$, and $\mathbf{N}$ that are explained by the noisy linear model (1). We estimate the values of the hidden variables by using MAP

$$p(\mathbf{A}, \mathbf{S}|\mathbf{X}) = \frac{p(\mathbf{X}|\mathbf{A}, \mathbf{S})p(\mathbf{A}, \mathbf{S})}{p(\mathbf{X})} = \frac{p_n(\mathbf{X}|\mathbf{A}, \mathbf{S})p_a(\mathbf{A})p_s(\mathbf{S})}{p(\mathbf{X})} \qquad (2)$$

with $p_a(\mathbf{A})$, $p_s(\mathbf{S})$, $p_n(\mathbf{N})$ as the a priori assumptions of the distributions. With MAP we estimate the most probable values for given priors after observing the data,

$$\mathbf{A}_{\text{MAP}}, \mathbf{S}_{\text{MAP}} = \arg\max_{\mathbf{A}, \mathbf{S}} p(\mathbf{A}, \mathbf{S}|\mathbf{X}) \qquad (3)$$

Note that for maximization the constant factor $p(\mathbf{X})$ can be ignored. Our first assumption, which is indicated in equation (2) is that the abundances are independent of the reflectance spectra as their origins are completely unrelated: (A0) $\mathbf{A}$ and $\mathbf{S}$ are independent.

The MAP algorithm is entirely defined by the choices of priors that are guided by the problem of hyperspectral unmixing: (A1) $\mathbf{A}$ represent probabilities for each pixel $i$. (A2) $\mathbf{S}$ are independent for different material $m$. (A3) $\mathbf{N}$ are normal i.i.d. for all $i, \lambda$. In summary, our MAP framework includes the assumptions A0-A3.

### 2.2    Including Priors

**Priors on the abundances**    Positivity and normalization of the abundances can be represented as,

$$p_a(\mathbf{A}) = \delta(\mathbf{A}\mathbf{1}_M - \mathbf{1}_N)\Theta(\mathbf{A}), \qquad (4)$$

where $\delta()$ represent the Kronecker delta function and $\Theta()$ the step function. With this choice a point not satisfying the constraint will have zero a posteriori probability. This prior introduces no particular bias of the solutions other then abundance constraints. It does however assume the abundances of different pixels to be independent.

**Prior on spectra**    Usually we find systematic trends in the spectra that cause significant correlation. However such an overall trend can be subtracted and/or filtered from the data leaving only independent signals that encode the variation from that overall trend. For example one can capture the conditional dependency structure with a linear auto-regressive (AR) model and analyze the resulting "innovations" or prediction errors [3]. In our model we assume that the spectra represent independent instances of an AR process having a white innovation process $e_{m\lambda}$ distributed according to $p_e(e)$. With a Toeplitz matrix $\mathbf{T}$ of the AR coefficients we

can write, $\mathbf{e}_m = \mathbf{s}_m \mathbf{T}$. The AR coefficients can be found in a preprocessing step on the observations $\mathbf{X}$. If $\mathbf{S}$ now represents the innovation process itself, our prior can be represented as,

$$p_e(\mathbf{S}) \propto p_e(\mathbf{ST}) = \prod_{m=1}^{M} \prod_{\lambda=1}^{L} p_e\left(\sum_{\lambda'=1}^{L} s_{m\lambda'} t_{\lambda\lambda'}\right), \tag{5}$$

Additionally $p_e(e)$ is parameterized by a mean and scale parameter and potentially parameters determining the higher moments of the distributions. For brevity we ignore the details of the parameterization in this paper.

**Prior on the noise** As outlined in the introduction there are a number of problems that can cause the linear model $\mathbf{X} = \mathbf{AS}$ to be inaccurate (e.g. multiple reflections, inhomogeneous atmospheric absorption, and detector noise.) As it is hard to treat all these phenomena explicitly, we suggest to pool them into one noise variable that we assume for simplicity to be normal distributed with a wavelength dependent noise variance $\sigma_\lambda$,

$$p(\mathbf{X}|\mathbf{A},\mathbf{S}) = p_n(\mathbf{N}) = \mathcal{N}(\mathbf{X} - \mathbf{AS}, \mathbf{\Sigma}) = \prod_{\lambda=1}^{L} \mathcal{N}(\mathbf{x}_\lambda - \mathbf{As}_\lambda, \sigma_\lambda \mathbf{I}), \tag{6}$$

where $\mathcal{N}(\cdot,\cdot)$ represents a zero mean Gaussian distribution, and $\mathbf{I}$ the identity matrix indicating the independent noise at each pixel.

### 2.3 MAP Solution for Zero Noise Case

Let us consider the noise-free case. Although this simplification may be inaccurate it will allow us to greatly reduce the number of free hidden variables - from $NM+ML$ to $M^2$. In the noise-free case the variables $\mathbf{A},\mathbf{S}$ are then deterministically dependent on each other through a $NL$-dimensional $\delta$-distribution, $p_n(\mathbf{X}|\mathbf{AS}) = \delta(\mathbf{X} - \mathbf{AS})$. We can remove one of these variables from our discussion by integrating (2). It is instructive to first consider removing $\mathbf{A}$

$$p(\mathbf{S}|\mathbf{X}) \propto \int d\mathbf{A}\, \delta(\mathbf{X} - \mathbf{AS}) p_a(\mathbf{A}) p_s(\mathbf{S}) = |\mathbf{S}^{-1}| p_a(\mathbf{XS}^{-1}) p_s(\mathbf{S}). \tag{7}$$

We omit tedious details and assume $L = M$ and invertible $\mathbf{S}$ so that we can perform the variable substitution that introduces the Jacobian determinant $|\mathbf{S}^{-1}|$. Let us consider the influence of the different terms. The Jacobian determinant measures the volume spanned by the endmembers $\mathbf{S}$. Maximizing its inverse will therefore try to shrink the simplex spanned by $\mathbf{S}$. The term $p_a(\mathbf{XS}^{-1})$ should guarantee that all data points map into the inside of the simplex, since the term should contribute zero or low probability for points that violate the constraint. Note that these two terms, in principle, define the same objective as the simplex envelope fitting algorithms previously mentioned [2].

In the present work we are more interested in the algorithm that results from removing $\mathbf{S}$ and finding the MAP estimate of $\mathbf{A}$. We obtain (cf. Eq.(7))

$$p(\mathbf{A}|\mathbf{X}) \propto \int d\mathbf{S}\, \delta(\mathbf{X} - \mathbf{AS}) p_a(\mathbf{A}) p_s(\mathbf{S}) = |\mathbf{A}^{-1}| p_s(\mathbf{A}^{-1}\mathbf{X}) p_a(\mathbf{A}). \tag{8}$$

For now we assumed $N = M$.[1] If $p_s(\mathbf{S})$ factors over $m$, i.e. endmembers are independent, maximizing the first two terms represents the ICA algorithm. However,

the prior on $\mathbf{A}$ will restrict the solutions to satisfy the abundance constraints and bias the result depending on the detailed choice of $p_a(\mathbf{A})$, so we are led to *constrained ICA*.

In summary, depending on which variable we integrate out we obtain two methods for solving the spectral unmixing problem: the known technique of simplex fitting and a new constrained ICA algorithm.

## 2.4  MAP Solution for the Noisy Case

Combining the choices for the priors made in section 2.2 (Eqs.(4), (5) and (6)) with (2) and (3) we obtain

$$
\mathbf{A}_{\mathrm{MAP}}, \mathbf{S}_{\mathrm{MAP}} = \arg\max_{\mathbf{A},\mathbf{S}} \prod_{\lambda=q}^{L} \left\{ \prod_{i=1}^{N} \mathcal{N}(x_{i\lambda} - \mathbf{a}_i \mathbf{s}_\lambda, \sigma_\lambda) \prod_{m=1}^{M} p_e \left( \sum_{\lambda'=1}^{L} s_{m\lambda'} t_{\lambda\lambda'} \right) \right\}, \quad (9)
$$

subject to $\mathbf{A}1_M = 1_L, \mathbf{A} \geq 0$. The logarithm of the cost function in (9) is denoted by $L = L(\mathbf{A}, \mathbf{S})$. Its gradient with respect to the hidden variables is

$$
\frac{\partial L}{\partial \mathbf{s}_m} = -\mathbf{A}^T \mathbf{n}_m \operatorname{diag}(\boldsymbol{\sigma})^{-1} - f_s(\mathbf{s}_m) \qquad (10)
$$

where $\mathbf{N} = \mathbf{X} - \mathbf{AS}$, $\mathbf{n}_m$ are the M column vectors of $\mathbf{N}$, $f_s(s) = -\frac{\partial \ln p_e(s)}{\partial s}$. In (10) $f_s$ is applied to each element of $\mathbf{s}_m$.

The optimization with respect to $\mathbf{A}$ for given $\mathbf{S}$ can be implemented as a standard *weighted least squares* (LS) problem with a linear constraint and positivity bounds. Since the constraints apply for every pixel independently one can solve $N$ separate constrained LS problems of $M$ unknowns each. We alternate between gradient steps for $\mathbf{S}$ and explicit solutions for $\mathbf{A}$ until convergence. Any additional parameters of $p_e(e)$ such as scale and mean may be obtained in a maximum likelihood (ML) sense by maximizing $L$. Note that the nonlinear optimization is not subject to constraints; the constraints apply only in the quadratic optimization.

# 3  Experiments

## 3.1  Zero Noise Case: Artificial Mixtures

In our first experiment we use mineral data from the United States Geological Survey (USGS)[2] to build artificial mixtures for evaluating our unsupervised unmixing framework. Three target endmembers where chosen (Almandine WS479, Montmorillonite+Illi CM42 and Dickite NMNH106242). A spectral scene of 100 samples was constructed by creating a random mixture of the three minerals. Of the 100 samples, there were no pure samples (i.e. no mineral had more than a 80% abundance in any sample). Figure 1A is the spectra of the endmembers recovered by the constrained ICA technique of section 2.3, where the constraints were implemented with penalty terms added to the conventional maximum likelihood ICA algorithm. These are nearly identical to the spectra of the true endmembers, shown in figure 1B, which were used for mixing. Interesting to note is the scatter-plot of the 100 samples across two bands. The open circles are the absorption values at these two bands for endmembers found by the MAP technique. Given that each mixed sample consists of no more than 80% of any endmember, the endmember points on the scatter-plot are quite distant from the cluster. A simplex fitting technique would have significant difficulty recovering the endmembers from this clustering.

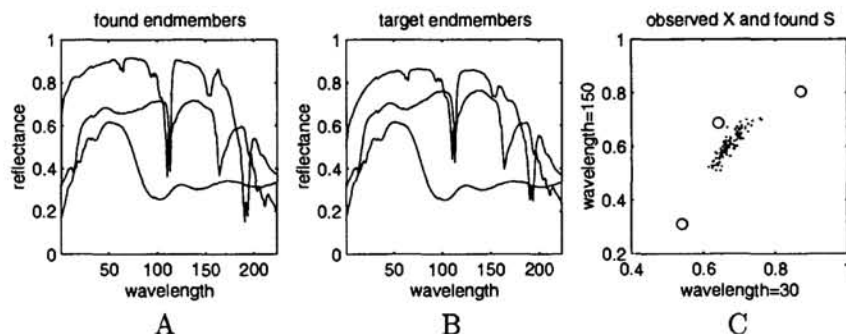

Figure 1: Results for noise-free artificial mixture. **A** recovered endmembers using MAP technique. **B** "true" target endmembers. **C** scatter plot of samples across 2 bands showing the absorption of the three endmembers computed by MAP (open circles).

## 3.2 Noisy Case: Real Mixtures

To validate the noise model MAP framework of section 2.4 we conducted an experiment using ground truthed USGS data representing real mixtures. We selected 10x10 blocks of pixels from three different regions[3] in the AVIRIS data of the Cuprite, Nevada mining district. We separate these 300 mixed spectra assuming two endmembers and an AR detrending with 5 AR coefficients and the MAP techniques of section 2.4. Overall brightness was accounted for as explain in the linear modeling of section 1. The endmembers are shown in figure 2A and B in comparison to laboratory spectra from the USGS spectral library for these minerals [8]. Figure 2C shows the corresponding abundances, which match the ground truth; region (III) mainly consists of Muscovite while regions (I)+(II) contain (areal) mixtures of Kaolinite and Muscovite.

## 4 Discussion

Hyperspectral unmixing is a challenging practical problem for unsupervised learning. Our probabilistic approach leads to several interesting algorithms: (1) simplex fitting, (2) constrained ICA and (3) constrained least squares that can efficiently use multi-channel information. An important element of our approach is the explicit use of prior information. Our simulation examples show that we can recover the endmembers, even in the presence of noise and model uncertainty. The approach described in this paper does not yet exploit local correlations between neighboring pixels that are well known to exist. Future work will therefore exploit not only *spectral* but also *spatial* prior information for detecting objects and materials.

**Acknowledgments**

We would like to thank Gregg Swayze at the USGS for assistance in obtaining the data.

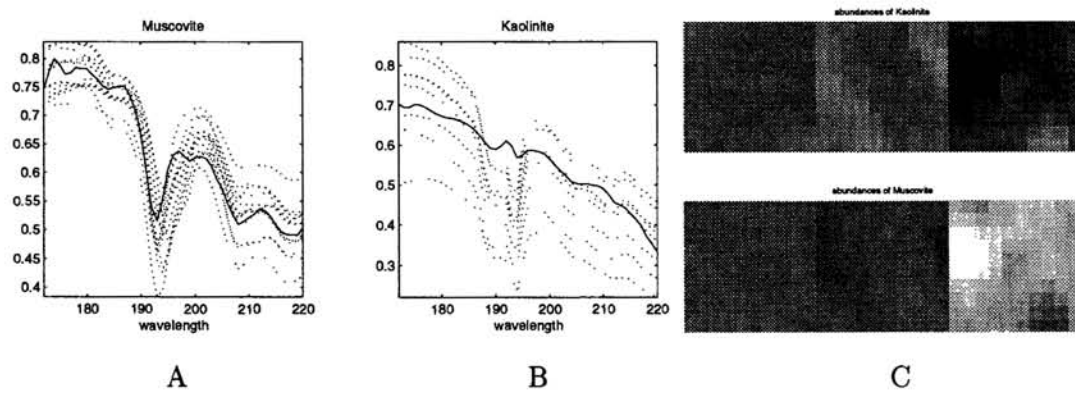

Figure 2: **A** Spectra of computed endmember (solid line) vs Muscovite sample spectra from the USGS data base library. Note we show only part of the spectrum since the discriminating features are located only between band 172 and 220. **B** Computed endmember (solid line) vs Kaolinite sample spectra from the USGS data base library. **C** Abundances for Kaolinite and Muscovite for three regions (lighter pixels represent higher abundance). Region 1 and region 2 have similar abundances for Kaolinite and Muscovite, while region 3 contains more Muscovite.

## Footnotes

[1] In practice more frequently we have $N > M$. In that case the observations $\mathbf{X}$ can be mapped into a $M$ dimensional subspace using the singular value decomposition (SVD), $\mathbf{X} = \mathbf{UDV}^T$. The discussion applies then to the reduced observations $\bar{\mathbf{X}} = \mathbf{U}_M^T \mathbf{X}$ with $\mathbf{U}_M$ being the first $M$ columns of $\mathbf{U}$.

[2]see http://speclab.cr.usgs.gov/spectral.lib.456.descript/decript04.html

[3]The regions were from the image plate2.cuprite95.alpha.2um.image.wlocals.gif in ftp://speclab.cr.usgs.gov/pub/cuprite/gregg.thesis.images/, at the coordinates (265,710) and (275,697), which contained Kaolinite and Muscovite 2, and (143,661), which only contained Muscovite 2.

# References

[1] J. Bayliss, J. A. Gualtieri, and R. Cromp. Analyzing hyperspectral data with independent component analysis. In J. M. Selander, editor, *Proc. SPIE Applied Image and Pattern Recognition Workshop*, volume 9, P.O. Box 10, Bellingham WA 98227-0010, 1997. SPIE.

[2] J.W. Boardman and F.A. Kruse. Automated spectral analysis: a geologic example using AVIRIS data, north Grapevine Mountains, Nevada. In *Tenth Thematic Conference on Geologic Remote Sensing*, pages 407–418, Ann arbor, MI, 1994. Environmental Research Institute of Michigan.

[3] S. Haykin. *Adaptive Filter Theory*. Prentice Hall, 1991.

[4] F. Maselli, , M. Pieri, and C. Conese. Automatic identification of end-members for the spectral decomposition of remotely sensed scenes. *Remote Sensing for Geography, Geology, Land Planning, and Cultural Heritage (SPIE)*, 2960:104–109, 1996.

[5] B. Pearlmutter and L. Parra. Maximum likelihood blind source separation: A context-sensitive generalization of ICA. In M. Mozer, M. Jordan, and T. Petsche, editors, *Advances in Neural Information Processing Systems 9*, pages 613–619, Cambridge MA, 1997. MIT Press.

[6] J.J. Settle. Linear mixing and the estimation of ground cover proportions. *International Journal of Remote Sensing*, 14:1159–1177, 1993.

[7] M.O. Smith, J.B. Adams, and A.R. Gillespie. Reference endmembers for spectral mixture analysis. In *Fifth Australian remote sensing conference*, volume 1, pages 331–340, 1990.

[8] U.S. Geological Survey. USGS digital spectral library. Open File Report 93-592, 1993.

[9] H. Szu and C. Hsu. Landsat spectral demixing a la superresolution of blind matrix inversion by constraint MaxEnt neural nets. In *Wavelet Applications IV*, volume 3078, pages 147–160. SPIE, 1997.
